# The Infinite Gamma-Poisson Feature Model

**Michalis K. Titsias**
School of Computer Science,
University of Manchester, UK
mtitsias@cs.man.ac.uk

## Abstract

We present a probability distribution over non-negative integer valued matrices with possibly an infinite number of columns. We also derive a stochastic process that reproduces this distribution over equivalence classes. This model can play the role of the prior in nonparametric Bayesian learning scenarios where multiple latent features are associated with the observed data and each feature can have multiple appearances or occurrences within each data point. Such data arise naturally when learning visual object recognition systems from unlabelled images. Together with the nonparametric prior we consider a likelihood model that explains the visual appearance and location of local image patches. Inference with this model is carried out using a Markov chain Monte Carlo algorithm.

## 1 Introduction

Unsupervised learning using mixture models assumes that one latent cause is associated with each data point. This assumption can be quite restrictive and a useful generalization is to consider factorial representations which assume that multiple causes have generated the data [11]. Factorial models are widely used in modern unsupervised learning algorithms; see e.g. algorithms that model text data [2, 3, 4]. Algorithms for learning factorial models should deal with the problem of specifying the size of the representation. Bayesian learning and especially nonparametric methods such as the Indian buffet process [7] can be very useful for solving this problem.

Factorial models usually assume that each feature occurs once in a given data point. This is inefficient to model the precise generation mechanism of several data such as images. An image can contain views of multiple object classes such as cars and humans and each class may have multiple occurrences in the image. To deal with features having multiple occurrences, we introduce a probability distribution over sparse non-negative integer valued matrices with possibly an unbounded number of columns. Each matrix row corresponds to a data point and each column to a feature similarly to the binary matrix used in the Indian buffet process [7]. Each element of the matrix can be zero or a positive integer and expresses the number of times a feature occurs in a specific data point. This model is derived by considering a finite gamma-Poisson distribution and taking the infinite limit for equivalence classes of non-negative integer valued matrices. We also present a stochastic process that reproduces this infinite model. This process uses the Ewens's distribution [5] over integer partitions which was introduced in population genetics literature and it is equivalent to the distribution over partitions of objects induced by the Dirichlet process [1].

The infinite gamma-Poisson model can play the role of the prior in a nonparametric Bayesian learning scenario where both the latent features and the number of their occurrences are unknown. Given this prior, we consider a likelihood model which is suitable for explaining the visual appearance and location of local image patches. Introducing a prior for the parameters of this likelihood model, we apply Bayesian learning using a Markov chain Monte Carlo inference algorithm and show results in some image data.

## 2 The finite gamma-Poisson model

Let $X = \{X_1, \ldots, X_N\}$ be some data where each data point $X_n$ is a set of attributes. In section 4 we specify $X_n$ to be a collection of local image patches. We assume that each data point is associated with a set of latent features and each feature can have multiple occurrences. Let $z_{nk}$ denote the number of times feature $k$ occurs in the data point $X_n$. Given $K$ features, $Z = \{z_{nk}\}$ is a $N \times K$ non-negative integer valued matrix that collects together all the $z_{nk}$ values so as each row corresponds to a data point and each column to a feature. Given that $z_{nk}$ is drawn from a Poisson with a feature-specific parameter $\lambda_k$, $Z$ follows the distribution

$$P(Z|\{\lambda_k\}) = \prod_{n=1}^{N} \prod_{k=1}^{K} \frac{\lambda_k^{z_{nk}} \exp\{-\lambda_k\}}{z_{nk}!} = \prod_{k=1}^{K} \frac{\lambda_k^{m_k} \exp\{-N\lambda_k\}}{\prod_{n=1}^{N} z_{nk}!}, \tag{1}$$

where $m_k = \sum_{n=1}^{N} z_{nk}$. We further assume that each $\lambda_k$ parameter follows a gamma distribution that favors sparsity (in a sense that will be explained shortly):

$$\mathcal{G}(\lambda_k; \frac{\alpha}{K}, 1) = \frac{\lambda_k^{\frac{\alpha}{K}-1} \exp\{-\lambda_k\}}{\Gamma(\frac{\alpha}{K})}. \tag{2}$$

The hyperparameter $\alpha$ itself is given a vague gamma prior $\mathcal{G}(\alpha; \alpha_0, \beta_0)$. Using the above equations we can easily integrate out the parameters $\{\lambda_k\}$ as follows

$$P(Z|\alpha) = \prod_{k=1}^{K} \frac{\Gamma(m_k + \frac{\alpha}{K})}{\Gamma(\frac{\alpha}{K})(N+1)^{m_k+\frac{\alpha}{K}} \prod_{n=1}^{N} z_{nk}!}, \tag{3}$$

which shows that given the hyperparameter $\alpha$ the columns of $Z$ are independent. Note that the above distribution is exchangeable since reordering the rows of $Z$ does not alter the probability. Also as $K$ increases the distribution favors sparsity. This can be shown by taking the expectation of the sum of all elements of $Z$. Since the columns are independent this expectation is $K \sum_{n=1}^{N} E(z_{nk})$ and $E(z_{nk})$ is given by

$$E(z_{nk}) = \sum_{z_{nk}=0}^{\infty} z_{nk} NB(z_{nk}; \frac{\alpha}{K}, \frac{1}{2}) = \frac{\alpha}{K}, \tag{4}$$

where $NB(z_{nk}; r, p)$, with $r > 0$ and $0 < p < 1$, denotes the negative binomial distribution over positive integers

$$NB(z_{nk}; r, p) = \frac{\Gamma(r + z_{nk})}{z_{nk}! \Gamma(r)} p^r (1-p)^{z_{nk}}, \tag{5}$$

that has a mean equal to $\frac{r(1-p)}{p}$. Using Equation (4) the expectation of the sum of $z_{nk}$s is $\alpha N$ and is independent of the number of features. As $K$ increases, $Z$ becomes sparser and $\alpha$ controls the sparsity of this matrix.

There is an alternative way of deriving the joint distribution $P(Z|\alpha)$ according to the following generative process:

$$(\theta_1, \ldots, \theta_K) \sim D\left(\frac{\alpha}{K}\right), \quad \lambda \sim \mathcal{G}(\lambda; \alpha, 1),$$

$$L_n \sim Poisson(\lambda), \quad (z_{n1}, \ldots, z_{nK}) \sim \binom{L_n}{z_{n1} \ldots z_{nK}} \prod_{k=1}^{K} \theta_k^{z_{nk}}, \quad n = 1, \ldots, N,$$

where $D(\frac{\alpha}{K})$ denotes the symmetric Dirichlet. Marginalizing out $\boldsymbol{\theta}$ and $\lambda$ gives rise to the same distribution $P(Z|\alpha)$. The above process generates a gamma random variable and multinomial parameters and then samples the rows of $Z$ independently by using the Poisson-multinomial pair. The connection with the Dirichlet-multinomial pair implies that the infinite limit of the gamma-Poisson model must be related to the Dirichlet process. In the next section we see how this connection is revealed through the Ewens's distribution [5].

Models that combine gamma and Poisson distributions are widely applied in statistics. We point out that the above finite model shares similarities with the techniques presented in [3, 4] that model text data.

# 3 The infinite limit and the stochastic process

To express the probability distribution in (3) for infinite many features $K$ we need to consider equivalence classes of $Z$ matrices similarly to [7]. The association of columns in $Z$ with features defines an arbitrary labelling of the features. Given that the likelihood $p(X|Z)$ is not affected by relabelling the features, there is an equivalence class of matrices that all can be reduced to the same standard form after column reordering. We define the left-ordered form of non-negative integer valued matrices as follows. We assume that for any possible $z_{nk}$ holds $z_{nk} \leq c - 1$, where $c$ is a sufficiently large integer. We define $h = (z_{1k} \ldots z_{Nk})$ as the integer number associated with column $k$ that is expressed in a numeral system with basis $c$. The left-ordered form is defined so as the columns of $Z$ appear from left to right in a decreasing order according to the magnitude of their numbers.

Starting from Equation (3) we wish to define the probability distribution over matrices constrained in a left-ordered standard form. Let $K_h$ be the multiplicity of the column with number $h$; for example $K_0$ is the number of zero columns. An equivalence class $[Z]$ consists of $\frac{K!}{\sum_{h=0}^{c^N-1} K_h!}$ different matrices that they are generated from the distribution in (3) with equal probabilities and can be reduced to the same left-ordered form. Thus, the probability of $[Z]$ is

$$P([Z]) = \frac{K!}{\sum_{h=0}^{c^N-1} K_h!} \prod_{k=1}^{K} \frac{\Gamma(m_k + \frac{\alpha}{K})}{\Gamma(\frac{\alpha}{K})(N+1)^{m_k + \frac{\alpha}{K}} \prod_{n=1}^{N} z_{nk}!}. \tag{6}$$

We assume that the first $K_+$ features are represented i.e. $m_k > 0$ for $k \leq K_+$, while the rest $K - K_+$ features are unrepresented i.e. $m_k = 0$ for $k > K_+$. The infinite limit of (6) is derived by following a similar strategy with the one used for expressing the distribution over partitions of objects as a limit of the Dirichlet-multinomial pair [6, 9]. The limit takes the following form:

$$P(Z|\alpha) = \frac{1}{\sum_{h=1}^{c^N-1} K_h!} \frac{\alpha^{K_+}}{(N+1)^{m+\alpha}} \frac{\prod_{k=1}^{K_+} (m_k - 1)!}{\prod_{k=1}^{K_+} \prod_{n=1}^{N} z_{nk}!}, \tag{7}$$

where $m = \sum_{k=1}^{K_+} m_k$. This expression defines an exchangeable joint distribution over non-negative integer valued matrices with infinite many columns in a left-ordered form. Next we present a sequential stochastic process that reproduces this distribution.

## 3.1 The stochastic process

The distribution in Equation (7) can be derived from a simple stochastic process that constructs the matrix $Z$ sequentially so as the data arrive one at each time in a fixed order. The steps of this stochastic process are discussed below.

When the first data point arrives all the features are currently unrepresented. We sample feature occurrences from the set of unrepresented features as follows. Firstly, we draw an integer number $g_1$ from the negative binomial $NB(g_1; \alpha, \frac{1}{2})$ which has a mean value equal to $\alpha$. $g_1$ is the total number of feature occurrences for the first data point. Given $g_1$, we randomly select a partition $(z_{11}, \ldots, z_{1K_1})$ of the integer $g_1$ into parts[1], i.e. $z_{11} + \ldots + z_{1K_1} = g_1$ and $1 \leq K_1 \leq g_1$, by drawing from Ewens's distribution [5] over integer partitions which is given by

$$P(z_{11}, \ldots, z_{1K_1}) = \alpha^{K_1} \frac{\Gamma(\alpha)}{\Gamma(g_1 + \alpha)} \frac{g_1!}{z_{11} \times \ldots \times z_{1K_1}} \prod_{i=1}^{g_1} \frac{1}{v_i^{(1)}!}, \tag{8}$$

where $v_i^{(1)}$ is the multiplicity of integer $i$ in the partition $(z_{11}, \ldots, z_{1K_1})$. The Ewens's distribution is equivalent to the distribution over partitions of objects induced by the Dirichlet process and the Chinese restaurant process since we can derive the one from the other using simple combinatorics arguments. The difference between them is that the former is a distribution over integer partitions while the latter is a distribution over partitions of objects.

Let $K_{n-1}$ be the number of represented features when the $n$th data point arrives. For each feature $k$, with $k \leq K_{n-1}$, we choose $z_{nk}$ based on the popularity of this feature in the previous $n-1$ data

points. This popularity is expressed by the total number of occurrences for the feature $k$ which is given by $m_k = \sum_{i=1}^{n-1} z_{ik}$. Particularly, we draw $z_{nk}$ from $NB(z_{nk}; m_k, \frac{n}{n+1})$ which has a mean value equal to $\frac{m_k}{n}$. Once we have sampled from all represented features we need to consider a sample from the set of unrepresented features. Similarly to the first data point, we first draw an integer $g_n$ from $NB(g_n; \alpha, \frac{n}{n+1})$, and subsequently we select a partition of that integer by drawing from the Ewens's formula. This process produces the following distribution:

$$P(Z|\alpha) = \frac{1}{\prod_{i=1}^{g_1} v_i^{(1)}! \times \ldots \times \prod_{i=1}^{g_N} v_i^{(N)}!} \frac{\alpha^{K_+}}{(N+1)^{m+\alpha}} \frac{\prod_{k=1}^{K_+}(m_k-1)!}{\prod_{k=1}^{K_+} \prod_{n=1}^{N} z_{nk}!}, \tag{9}$$

where $\{v_i^{(n)}\}$ are the integer-multiplicities for the $n$th data point which arise when we draw from the Ewens's distribution. Note that the above expression does not have exactly the same form as the distribution in Equation (7) and is not exchangeable since it depends on the order the data arrive. However, if we consider only the left-ordered class of matrices generated by the stochastic process then we obtain the exchangeable distribution in Equation (7). Note that a similar situation arises with the Indian buffet process.

## 3.2 Conditional distributions

When we combine the prior $P(Z|\alpha)$ with a likelihood model $p(X|Z)$ and we wish to do inference over $Z$ using Gibbs-type sampling, we need to express the conditionals of the form $P(z_{nk}|Z_{-(nk)}, \alpha)$ where $Z_{-(nk)} = Z \setminus z_{nk}$. We can derive such conditionals by taking limits of the conditionals for the finite model or by using the stochastic process.

Suppose that for the current value of $Z$, there exist $K_+$ represented features i.e. $m_k > 0$ for $k \leq K_+$. Let $m_{-n,k} = \sum_{\tilde{n} \neq n} z_{\tilde{n}k}$. When $m_{-n,k} > 0$, the conditional of $z_{nk}$ is given by $NB(z_{nk}; m_{-n,k}, \frac{N}{N+1})$. In all different cases, we need a special conditional that samples from new features[2] and accounts for all $k$ such that $m_{-n,k} = 0$. This conditional draws an integer number from $NB(g_n; a, \frac{N}{N+1})$ and then determines the occurrences for the new features by choosing a partition of the integer $g_n$ using the Ewens's distribution. Finally the conditional $p(\alpha|Z)$, which can be directly expressed from Equation (7) and the prior of $\alpha$, is given by

$$p(\alpha|Z) \propto \mathcal{G}(\alpha; \alpha_0, \beta_0) \frac{\alpha^{K_+}}{(N+1)^{\alpha}}. \tag{10}$$

Typically the likelihood model does not depend on $\alpha$ and thus the above quantity is also the posterior conditional of $\alpha$ given data and $Z$.

## 4   A likelihood model for images

An image can contain multiple objects of different classes. Each object class can have more than one occurrences, i.e. multiple instances of the class may appear simultaneously in the image. Unsupervised learning should deal with the unknown number of object classes in the images and also the unknown number of occurrences of each class in each image separately. If object classes are the latent features, what we wish to infer is the underlying feature occurrence matrix $Z$. We consider an observation model that is a combination of latent Dirichlet allocation [2] and Gaussian mixture models. Such a combination has been used before [12]. Each image $n$ is represented by $d_n$ local patches that are detected in the image so as $X_n = (Y_n, W_n) = \{(\mathbf{y}_{ni}, \mathbf{w}_{ni}), i = 1, \ldots, d_n\}$. $\mathbf{y}_{ni}$ is the two-dimensional location of patch $i$ and $\mathbf{w}_{ni}$ is an indicator vector (i.e. is binary and satisfies $\sum_{\ell=1}^{L} w_{ni}^\ell = 1$) that points into a set of $L$ possible visual appearances. $X, Y$, and $W$ denote all the data the locations and the appearances, respectively. We will describe the probabilistic model starting from the joint distribution of all variables which is given by

$$\text{joint} = p(\alpha)P(Z|\alpha)p(\{\boldsymbol{\theta}_k\}|Z) \times$$
$$\prod_{n=1}^{N}\left[p(\boldsymbol{\pi}_n|Z_n)p(\mathbf{m}_n, \Sigma_n|Z_n)\prod_{i=1}^{d_n} P(\mathbf{s}_{ni}|\boldsymbol{\pi}_n)P(\mathbf{w}_{ni}|\mathbf{s}_{ni}, \{\boldsymbol{\theta}_k\})p(\mathbf{y}_{ni}|\mathbf{s}_{ni}, \mathbf{m}_n, \Sigma_n)\right]. \tag{11}$$

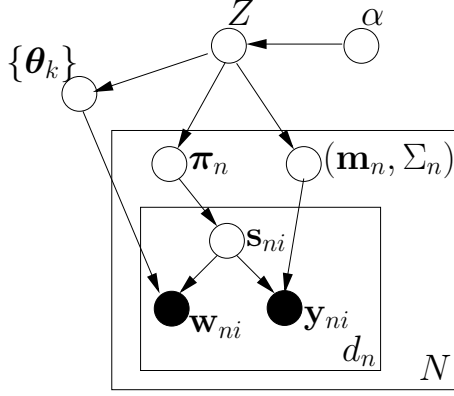

Figure 1: Graphical model for the joint distribution in Equation (11).

The graphical representation of this distribution is depicted in Figure 1. We now explain all the pieces of this joint distribution following the causal structure of the graphical model. Firstly, we generate $\alpha$ from its prior and then we draw the feature occurrence matrix $Z$ using the infinite gamma-Poisson prior $P(Z|\alpha)$. The matrix $Z$ defines the structure for the remaining part of the model. The parameter vector $\boldsymbol{\theta}_k = \{\theta_{k1}, \ldots, \theta_{kL}\}$ describes the appearance of the local patches $W$ for the feature (object class) $k$. Each $\boldsymbol{\theta}_k$ is generated from a symmetric Dirichlet so as the whole set of $\{\boldsymbol{\theta}_k\}$ vectors is drawn from $p(\{\boldsymbol{\theta}_k\}|Z) = \prod_{k=1}^{K_+} D(\boldsymbol{\theta}_k|\gamma)$, where $\gamma$ is the hyperparameter of the symmetric Dirichlet and it is common for all features. Note that the feature appearance parameters $\{\boldsymbol{\theta}_k\}$ depend on $Z$ only through the number of represented features $K_+$ which is obtained by counting the non-zero columns of $Z$.

The parameter vector $\boldsymbol{\pi}_n = \{\pi_{nkj}\}$ defines the image-specific mixing proportions for the mixture model associated with image $n$. To see how this mixture model arises, notice that a local patch in image $n$ belongs to a certain occurrence of a feature. We use the double index $kj$ to denote the $j$ occurrence of feature $k$ where $j = 1, \ldots, z_{nk}$ and $k \in \{\widetilde{k} : z_{n\widetilde{k}} > 0\}$. This mixture model has $M_n = \sum_{k=1}^{K_+} z_{nk}$ components, i.e. as many as the total number of feature occurrences in image $n$. The assignment variable $\mathbf{s}_{ni} = \{s_{ni}^{kj}\}$, which takes $M_n$ values, indicates the feature occurrence of patch $i$. $\boldsymbol{\pi}_n$ is drawn from a symmetric Dirichlet given by $p(\boldsymbol{\pi}_n|Z_n) = D(\boldsymbol{\pi}_n|\beta/M_n)$, where $Z_n$ denotes the $n$th row of $Z$ and $\beta$ is a hyperparameter shared by all images. Notice that $\boldsymbol{\pi}_n$ depends only on the $n$th row of $Z$.

The parameters $(\mathbf{m}_n, \Sigma_n)$ determine the image-specific distribution for the locations $\{\mathbf{y}_{ni}\}$ of the local patches in image $n$. We assume that each occurrence of a feature forms a Gaussian cluster of patch locations. Thus $\mathbf{y}_{ni}$ follows a image-specific Gaussian mixture with $M_n$ components. We assume that the component $kj$ has mean $\mathbf{m}_{nkj}$ and covariance $\Sigma_{nkj}$. $\mathbf{m}_{nkj}$ describes object location and $\Sigma_{nkj}$ object shape. $\mathbf{m}_n$ and $\Sigma_n$ collect all the means and covariances of the clusters in the image $n$. Given that any object can be anywhere in the image and have arbitrary scale and orientation, $(\mathbf{m}_{nkj}, \Sigma_{nkj})$ should be drawn from a quite vague prior. We use a conjugate normal-Wishart prior for the pair $(\mathbf{m}_{nkj}, \Sigma_{nkj})$ so as

$$p(\mathbf{m}_n, \Sigma_n|Z_n) = \prod_{k:z_{nk}>0} \prod_{j=1}^{z_{nk}} N(\mathbf{m}_{nkj}|\boldsymbol{\mu}, \tau\Sigma_{nkj}) W(\Sigma_{nkj}^{-1}|v, V), \qquad (12)$$

where $(\boldsymbol{\mu}, \tau, v, V)$ are the hyperparameters shared by all features and images. The assignment $\mathbf{s}_{ni}$ which determines the allocation of a local patch in a certain feature occurrence follows a multinomial: $P(s_{ni}|\boldsymbol{\pi}_n) = \prod_{k:z_{nk}>0} \prod_{j=1}^{z_{nk}} (\pi_{nkj})^{s_{ni}^{kj}}$. Similarly the observed data pair $(\mathbf{w}_{ni}, \mathbf{y}_{ni})$ of a local image patch is generated according to

$$P(\mathbf{w}_{ni}|\mathbf{s}_{ni}, \{\boldsymbol{\theta}_k\}) = \prod_{k=1}^{K_+} \prod_{\ell=1}^{L} \theta_{k\ell}^{w_{ni}^\ell \sum_{j=1}^{z_{nk}} s_{ni}^{kj}}$$

and

$$p(\mathbf{y}_{ni}|\mathbf{s}_{ni}, \mathbf{m}_n, \Sigma_n) = \prod_{k:z_{nk}>0} \prod_{j=1}^{z_{nk}} [N(\mathbf{y}_{ni}|\mathbf{m}_{nkj}, \Sigma_{nkj})]^{s_{ni}^{kj}}.$$

The hyperparameters $(\gamma, \beta, \boldsymbol{\mu}, \tau, v, V)$ take fixed values that give vague priors and they are not depicted in the graphical model shown in Figure 1.

Since we have chosen conjugate priors, we can analytically marginalize out from the joint distribution all the parameters $\{\boldsymbol{\pi}_n\}$, $\{\boldsymbol{\theta}_k\}$, $\{\mathbf{m}_n\}$ and $\{\Sigma_n\}$ and obtain $p(X, S, Z, \alpha)$. Marginalizing out the assignments $S$ is generally intractable and the MCMC algorithm discussed next produces samples from the posterior $P(S, Z, \alpha|X)$.

## 4.1 MCMC inference

Inference with our model involves expressing the posterior $P(S, Z, \alpha|X)$ over the feature occurrences $Z$, the assignments $S$ and the parameter $\alpha$. Note that the joint $P(S, Z, \alpha, X)$ factorizes according to $p(\alpha)P(Z|\alpha)P(W|S, Z) \prod_{n=1}^{N} P(S_n|Z_n)p(Y_n|S_n, Z_n)$ where $S_n$ denotes the assignments associated with image $n$. Our algorithm uses mainly Gibbs-type sampling from conditional posterior distributions. Due to space limitations we briefly discuss the main points of this algorithm.

The MCMC algorithm processes the rows of $Z$ iteratively and updates its values. A single step can change an element of $Z$ by one so as $|z_{nk}^{new} - z_{nk}^{old}| \leq 1$. Initially $Z$ is such that $M_n = \sum_{k=1}^{K_+} z_{nk} \geq 1$, for any $n$ which means that at least one mixture component explains the data of each image. The proposal distribution for changing $z_{nk}$s ensures that this constraint is satisfied.

Suppose we wish to sample a new value for $z_{nk}$ using the joint model $p(S, Z, \alpha, X)$. Simply witting $P(z_{nk}|S, Z_{-(nk)}, \alpha, X)$ is not useful since when $z_{nk}$ changes the number of states the assignments $S_n$ can take also changes. This is clear since $z_{nk}$ is a structural variable that affects the number of components $M_n = \sum_{k=1}^{K_+} z_{nk}$ of the mixture model associated with image $n$ and assignments $S_n$. On the other hand the dimensionality of the assignments $S_{-n} = S \setminus S_n$ of all other images is not affected when $z_{nk}$ changes. To deal with the above we marginalize out $S_n$ and we sample $z_{nk}$ from the marginalized posterior conditional $P(z_{nk}|S_{-n}, Z_{-(nk)}, \alpha, X)$ which is computed according to

$$P(z_{nk}|S_{-n}, Z_{-(nk)}, \alpha, X) \propto P(z_{nk}|Z_{-(nk)}, \alpha) \sum_{S_n} P(W|S, Z)p(Y_n|S_n, Z_n)P(S_n|Z_n), \quad (13)$$

where $P(z_{nk}|Z_{-n,k}, \alpha)$ for the infinite case is computed as described in section 3.2 while computing the sum requires an approximation. This sum is a marginal likelihood and we apply importance sampling using as an importance distribution the posterior conditional $P(S_n|S_{-n}, Z, W, Y_n)$ [10]. Sampling from $P(S_n|S_{-n}, Z, W, Y_n)$ is carried out by applying local Gibbs sampling moves and global Metropolis moves that allow two occurrences of different features to exchange their data clusters. In our implementation we consider a single sample drawn from this posterior distribution so that the sum is approximated by $P(W|S_n^*, S_{-n}, Z)p(Y_n|S_n^*, Z_n)$ and $S_n^*$ is a sample accepted after a burn in period. Additionally to scans that update $Z$ and $S$ we add few Metropolis-Hastings steps that update the hyperparameter $\alpha$ using the posterior conditional given by Equation (10).

## 5 Experiments

In the first experiment we use a set of 10 artificial images. We consider four features that have the regular shapes shown in Figure 2. The discrete patch appearances correspond to pixels and can take 20 possible grayscale values. Each feature has its own multinomial distribution over the appearances. To generate an image we first decide to include each feature with probability 0.5. Then for each included feature we randomly select the number of occurrences from the range $[1, 3]$. For each feature occurrence we select the pixels using the appearance multinomial and place the respective feature shape in a random location so that feature occurrences do not occlude each other. The first row of Figure 2 shows a training image (left), the locations of pixels (middle) and the discrete appearances (right). The MCMC algorithm was initialized with $K_+ = 1$, $\alpha = 1$ and $z_{n1} = 1$, $n = 1, \ldots, 10$. The third row of Figure 2 shows how $K_+$ (left) and the sum of all $z_{nk}$s (right) evolve through the first 500 MCMC iterations. The algorithm in the first 20 iterations has

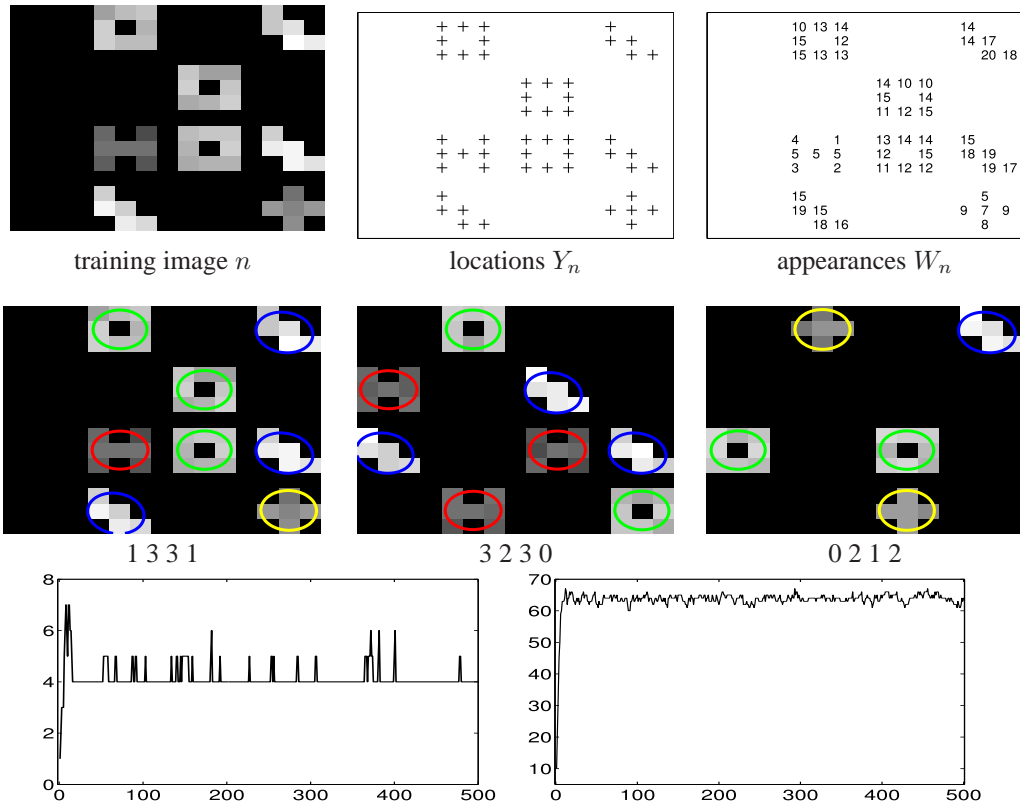

Figure 2: The first row shows a training image (left), the locations of pixels (middle) and the discrete appearances (right). The second row shows the localizations of all feature occurrences in three images. Below of each image the corresponding row of $Z$ is also shown. The third row shows how $K_+$ (left) and the sum of all $z_{nk}$s (right) evolve through the first 500 MCMC iterations.

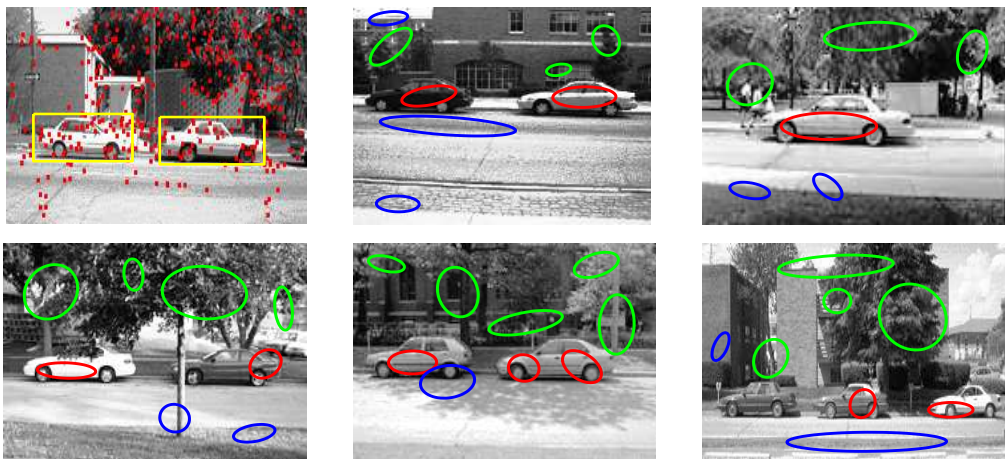

Figure 3: The left most plot on the first row shows the locations of detected patches and the bounding boxes in one of the annotated images. The remaining five plots show examples of detections and localizations of the three most dominant features (including the car-category) in five non-annotated images.

visited the matrix $Z$ that was used to generate the data and then stabilizes. For $86\%$ of the samples $K_+$ is equal to four. For the state $(Z, S)$ that is most frequently visited, the second row of Figure 2 shows the localizations of all different feature occurrences in three images. Each ellipse is drawn using the posterior mean values for a pair $(\mathbf{m}_{nkj}, \Sigma_{nkj})$ and illustrates the predicted location and shape of a feature occurrence. Note that ellipses with the same color correspond to the different occurrences of the same feature.

In the second experiment we consider 25 real images from the UIUC[3] cars database. We used the patch detection method presented in [8] and we constructed a dictionary of 200 visual appearances by clustering the SIFT [8] descriptors of the patches using K-means. Locations of detected patches are shown in the first row (left) of Figure 3. We partially labelled some of the images. Particularly, for 7 out of 25 images we annotated the car views using bounding boxes (Figure 3). This allows us to specify seven elements of the first column of the matrix Z (the first feature will correspond to the car-category). These $z_{nk}$s values plus the assignments of all patches inside the boxes do not change during sampling. Also the patches that lie outside the boxes in all annotated images are not allowed to be part of car occurrences. This is achieved by applying partial Gibbs sampling updates and Metropolis moves when sampling the assignments $S$. The algorithm is initialized with $K_+ = 1$, after 30 iterations stabilizes and then fluctuates between nine to twelve features. To keep the plots uncluttered, Figure 3 shows the detections and localizations of only the three most dominant features (including the car-category) in five non-annotated images. The red ellipses correspond to different occurrences of the car-feature, the green ones to a tree-feature and the blue ones to a street-feature.

## 6  Discussion

We presented the infinite gamma-Poisson model which is a nonparametric prior for non-negative integer valued matrices with infinite number of columns. We discussed the use of this prior for unsupervised learning where multiple features are associated with our data and each feature can have multiple occurrences within each data point. The infinite gamma-Poisson prior can be used for other purposes as well. For example, an interesting application can be Bayesian matrix factorization where a matrix of observations is decomposed into a product of two or more matrices with one of them being a non-negative integer valued matrix.

## Footnotes

[1] The partition of a positive integer is a way of writing this integer as a sum of positive integers where order does not matter, e.g. the partitions of 3 are: (3),(2,1) and (1,1,1).

[2]Features of this kind are the unrepresented features ($k > K_+$) as well as all the unique features that occur only in the data point $n$ (i.e. $m_{-n,k} = 0$, but $z_{nk} > 0$).

[3]available from `http://l2r.cs.uiuc.edu/~cogcomp/Data/Car/`.

## References

[1] C. Antoniak. Mixture of Dirichlet processes with application to Bayesian nonparametric problems. *The Annals of Statistics*, 2:1152–1174, 1974.

[2] D. M. Blei, A. Y. Ng, and M. I. Jordan. Latent Dirichlet allocation. *JMLR*, 3, 2003.

[3] W. Buntime and A. Jakulin. Applying discrete PCA in data analysis. In *UAI*, 2004.

[4] J. Canny. GaP: A factor model for discrete data. In *SIGIR*, pages 122–129. ACM Press, 2004.

[5] W. Ewens. The sampling theory of selectively neutral alleles. *Theoretical Population Biology*, 3:87–112, 1972.

[6] P. Green and S. Richardson. Modelling heterogeneity with and without the Dirichlet process. *Scandinavian Journal of Statistics*, 28:355–377, 2001.

[7] T. Griffiths and Z. Ghahramani. Infinite latent feature models and the Indian buffet process. In *NIPS 18*, 2006.

[8] D. G. Lowe. Distinctive image features from scale-invariant keypoints. *International Journal of Computer Vision*, 60(2):91–110, 2004.

[9] R. M. Neal. Bayesian mixture modeling. In *11th International Workshop on Maximum Entropy and Bayesian Methods of Statistical Analysis*, pages 197–211, 1992.

[10] M. A. Newton and A. E Raftery. Approximate Bayesian inference by the weighted likelihood bootstrap. *Journal of the Royal Statistical Society, Series B*, 3:3–48, 1994.

[11] E. Saund. A multiple cause mixture model for unsupervised learning. *Neural Computation*, 7:51–71, 1995.

[12] E. Sudderth, A. Torralba, W. T. Freeman, and A. Willsky. Describing Visual Scenes using Transformed Dirichlet Processes. In *NIPS 18*, 2006.

